# Generalized Hopfield Networks
# and
# Nonlinear Optimization

**Gintaras V. Reklaitis**
Dept. of Chemical Eng.
Purdue University
W. Lafayette, IN. 47907

**Athanasios G. Tsirukis**[1]
Dept. of Chemical Eng.
Purdue University
W. Lafayette, IN. 47907

**Manoel F. Tenorio**
Dept. of Electrical Eng.
Purdue University
W. Lafayette, IN. 47907

## ABSTRACT

A nonlinear neural framework, called the Generalized Hopfield network, is proposed, which is able to solve in a parallel distributed manner systems of nonlinear equations. The method is applied to the general nonlinear optimization problem. We demonstrate GHNs implementing the three most important optimization algorithms, namely the Augmented Lagrangian, Generalized Reduced Gradient and Successive Quadratic Programming methods. The study results in a dynamic view of the optimization problem and offers a straightforward model for the parallelization of the optimization computations, thus significantly extending the practical limits of problems that can be formulated as an optimization problem and which can gain from the introduction of nonlinearities in their structure (eg. pattern recognition, supervised learning, design of content-addressable memories).

## 1 RELATED WORK

The ability of networks of highly interconnected simple nonlinear analog processors (neurons) to solve complicated optimization problems was demonstrated in a series of papers by Hopfield and Tank (Hopfield, 1984), (Tank, 1986).

The Hopfield computational model is almost exclusively applied to the solution of combinatorially complex linear decision problems (eg. Traveling Salesman Problem). Unfortunately such problems can not be solved with guaranteed quality, (Bruck, 1987), getting trapped in locally optimal solutions.

Jeffrey and Rossner, (Jeffrey, 1986), extended Hopfield's technique to the nonlinear unconstrained optimization problem, using Cauchy dynamics. Kennedy and Chua, (Kennedy, 1988), presented an analog implementation of a network solving a nonlinear optimization problem. The underlying optimization algorithm is a simple transformation method, (Reklaitis, 1983), which is known to be relatively inefficient for large nonlinear optimization problems.

## 2 LINEAR HOPFIELD NETWORK (LHN)

The computation in a Hopfield network is done by a collection of highly interconnected simple neurons. Each processing element, i, is characterized by the activation level, $u_i$, which is a function of the input received from the external environment, $I_i$, and the state of the other neurons. The activation level of $i$ is transmitted to the other processors, after passing through a filter that converts $u_i$ to a 0-1 binary value, $V_i$.

The time behavior of the system is described by the following model:

$$C_i(\frac{du_i}{dt}) = \sum_j T_{ij}V_j - \frac{u_i}{R_i} + I_i$$

where $T_{ij}$ are the interconnection strengths. The network is characterized as linear, because the neuron inputs appear linearly in the neuron's constitutive equation. The steady-state of a Hopfield network corresponds to a local minimum of the corresponding quadratic Lyapunov function:

$$E = -\frac{1}{2}\sum_i \sum_j T_{ij}V_1V_j + \sum_i I_iV_i + \sum_i (\frac{1}{R_i}) \int_0^{V_i} s_i^{-1}(V)dV$$

If the matrix $[T_{ij}]$ is symmetric, the steady-state values of $V_i$ are binary These observations turn the Hopfield network to a very useful discrete optimization tool. Nonetheless, the linear structure poses two major limitations: The Lyapunov (objective) function can only take a quadratic form, whereas the feasible region can only have a hypercube geometry ($-1 \le V_i \le 1$). Therefore, the Linear Hopfield Network is limited to solve optimization problems with quadratic objective function and linear constraints. The general nonlinear optimization problem requires arbitrarily nonlinear neural interactions.

## 3 THE NONLINEAR OPTIMIZATION PROBLEM

The general nonlinear optimization problem consists of a search for the values of the independent variables $x_i$, optimizing a multivariable objective function so that some conditions (equality, $h_i$, and inequality, $g_j$, constraints) are satisfied at the optimum.

$$optimize \quad f(x_1, x_2, ..., x_n)$$

$$subject \quad to$$

$$h_i(x_1, x_2, ..., x_n) = 0 \qquad i = 1,2,...,K, \quad K < N$$

$$a_j \leq g_j(x_1, x_2, ..., x_n) \leq b_j \qquad j = 1,2,...,M$$

$$x_k^L \leq x_k \leq x_k^U \qquad k = 1,2,...,N$$

The influence of the constraint geometry on the shape of the objective function is described in a unified manner by the Lagrangian Function:

$$L = f - v^T h$$

The $v_i$ variables , also known as Lagrange multipliers, are unknown weighting parameters to be specified. In the optimum, the following conditions are satisfied:

$$\nabla_x L = 0 \qquad (N \ equations) \qquad (1)$$

$$\nabla_v L = 0 \qquad (K \ equations) \qquad (2)$$

From (1) and (2) it is clear that the optimization problem is transformed into a nonlinear equation solving problem. In a Generalized Hopfield Network each neuron represents an independent variable. The nonlinear connectivity among them is determined by the specific problem at hand and the implemented optimization algorithm. The network is designed to relax from an initial state to a steady-state that corresponds to a locally optimal solution of the problem.

Therefore, the optimization algorithms must be transformed into a dynamic model - system of differential equations - that will dictate the nonlinear neural interactions.

## 4 OPTIMIZATION METHODS

Cauchy and Newton dynamics are the two most important unconstrained optimization (equation solving) methods, adopted by the majority of the existing algorithms.

### 4.1 CAUCHY'S METHOD

This is the famous steepest descent algorithm, which tracks the direction of the largest change in the value of the objective function, $f$. The "equation of motion" for a Cauchy dynamic system is:

$$\frac{dx}{dt} = -\nabla f \quad ; \quad x(0) = x_o$$

## 4.2    NEWTON'S METHOD

If second-order information is available, a more rapid convergence is produced using Newton's approximation:

$$\frac{dx}{dt} = \pm (\nabla^2 f)^{-1} \nabla f \quad ; \quad x(0) = x_o$$

The steepest descent dynamics are very efficient initially, producing large objective-value changes, but close to the optimum they become very small, significantly increasing the convergence time. In contrast, Newton's method has a fast convergence close to the optimum, but the optimization direction is uncontrollable. The Levenberg - Marquardt heuristic, (Reklaitis, 1983), solves the problem by adopting Cauchy dynamics initially and switch to Newton dynamics near the optimum. Figure 1 shows the optimization trajectory of a Cauchy network. The algorithm converges to locally optimal solutions.

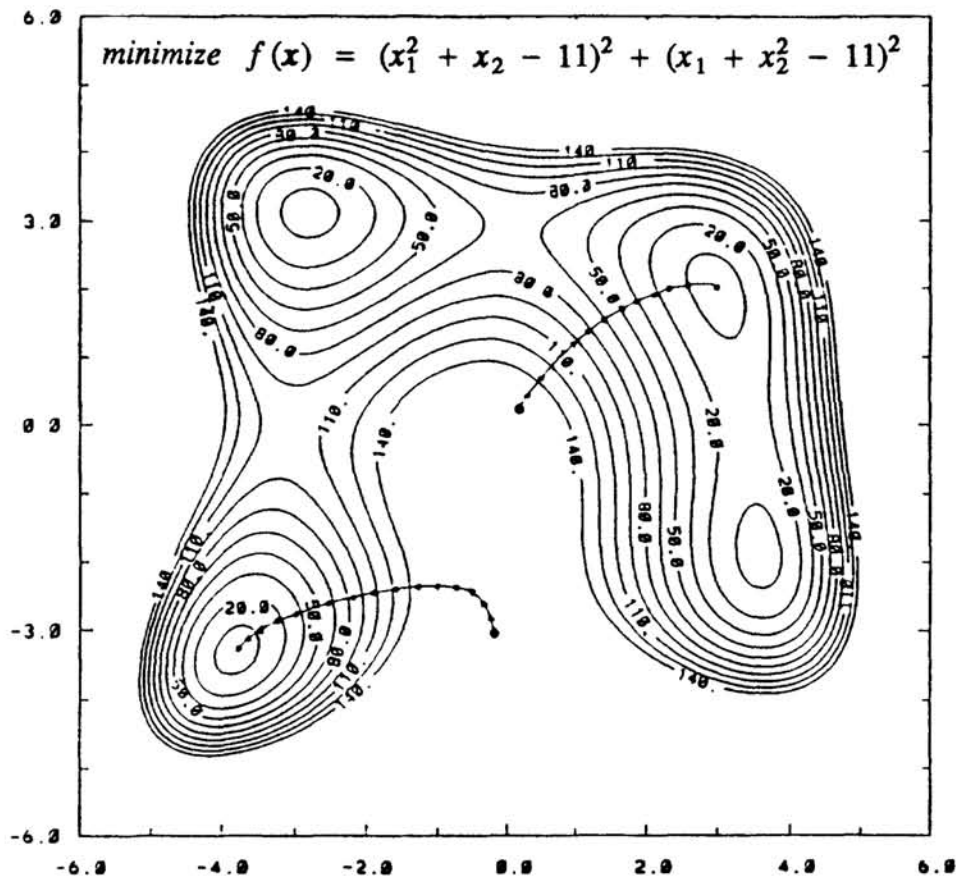

$$minimize \quad f(x) = (x_1^2 + x_2 - 11)^2 + (x_1 + x_2^2 - 11)^2$$

**Figure 1:** Convergence to Local Optima

# 5 CONSTRAINED OPTIMIZATION

The constrained optimization algorithms attempt to conveniently manipulate the equality and inequality constraints so that the problem is finally reduced to an unconstrained optimization, which is solved using Cauchy's or Newton's methods. Three are the most important constrained optimization algorithms: The Augmented Lagrangian, the Generalized Reduced Gradient (GRG) and the Successive Quadratic Programming (SQP). Corresponding Generalized Hopfield Networks will be developed for all of them.

## 5.1  TRANSFORMATION METHODS - AUGMENTED LAGRANGIAN

According to the transformation methods, a measure of the distance from the feasibility region is attached to the objective function and the problem is solved as an unconstrained optimization one. A transformation method was employed by Hopfield. These algorithms are proved inefficient because of numerical difficulties implicitly embedded in their structure, (Reklaitis, 1983). The Augmented Lagrangian is specifically designed to avoid these problems. The transformed unconstrained objective function becomes:

$$P(x,\sigma,\tau) = f(x) + R \sum_j \{ <g_j(x) + \sigma_j>^2 - \sigma_j^2 \}$$
$$+ R \sum_i \{ [h_i(x) + \tau_i]^2 - \tau_i^2 \}$$

where R is a predetermined weighting factor, and $\sigma_j$, $\tau_i$ the corresponding inequality - equality Lagrange multipliers. The operator $<a>$ returns $a$ for $a \le 0$. Otherwise it returns 0.

The design of an Augmented Lagrangian GHN requires $(N+K)$ neurons, where $N$ is the number of variables and $K$ is the number of constraints. The neuron connectivity of a GHN with Cauchy performance is described by the following model:

$$\frac{dx}{dt} = -\nabla_x P = -\nabla f - 2R <g + \sigma>^T \nabla g - 2R [h + \tau]^T \nabla h$$

$$\frac{d\sigma}{dt} = +\nabla_\sigma P = 2R <g + \sigma> - 2R \sigma$$

$$\frac{d\tau}{dt} = +\nabla_\tau P = 2R h$$

where $\nabla g$ and $\nabla h$ are matrices, eg. $\nabla h = [\nabla h_1, ..., \nabla h_k]$.

## 5.2  GENERALIZED REDUCED GRADIENT

According to the GRG method, $K$ variables (**basics**, $\hat{x}$) are determined by solving the $K$ nonlinear constraint equations, as functions of the rest $(N-K)$ variables (**non-basics**, $\bar{x}$). Subsequently the problem is solved as a reduced-dimension unconstrained optimization problem. Equations (1) and (2) are transformed to:

$$\nabla \tilde{f} = \nabla \overline{f} - \nabla \hat{f} \, (\nabla \hat{h})^{-1} \, \nabla \overline{h} = 0$$

$$h(x) = 0$$

The constraint equations are solved using Newton's method. Note that the Lagrange multipliers are explicitly eliminated. The design of a GRG GHN requires $N$ neurons, each one representing an independent variable. The neuron connectivity using Cauchy dynamics for the unconstrained optimization is given by:

$$\frac{d\overline{x}}{dt} = -\nabla \tilde{f} = -\nabla \overline{f} + \nabla \hat{f} \, (\nabla \hat{h})^{-1} \, \nabla \overline{h} \qquad (3)$$

$$h(x) = 0 \quad ( \rightarrow \frac{d\hat{x}}{dt} = h \, (\nabla \hat{h})^{-1} ) \qquad (4)$$

$$x(0) = x_0$$

System (3)-(4) is a differential - algebraic system, with an inherent sequential character: for each small step towards lower objective values, produced by (3), the system of nonlinear constraints should be solved, by relaxing equations (4) to a steady-state. The procedure is repeated until both equations (3) and (4) reach a steady state.

## 5.3    SUCCESSIVE QUADRATIC PROGRAMMING

In the SQP algorithm equations (1) and (2) are simultaneously solved as a nonlinear system of equations with both the independent variables, $x$, and the Lagrange multipliers, $v$, as unknowns. The solution is determined using Newton's method.

The design of an SQP GHN requires $(N+K)$ neurons representing the independent variables and the Lagrange multipliers. The connectivity of the network is determined by the following state equations:

$$\frac{dz}{dt} = \pm [\nabla^2 L]^{-1} \, (\nabla L)$$

$$z(0) = z_0$$

where $z$ is the augmented set of independent variables:

$$z = [x;v]$$

## 5.4    COMPARISON OF THE NETWORKS

The Augmented Lagrangian network is very easily programmed. Newton dynamics should be used very carefully because the operator $<a>$ is not smooth at $a = 0$.

The GRG network requires $K$ fewer neurons compared to the other networks. It requires more programming effort because of the inversion of the constraint Jacobian.

The SQP network is algorithmically the most effective, because second order information is used in the determination of both the variables and the multipliers. It is the most tedious to program because of the inversion of the Lagrange Hessian. All the GHNs are proved to be stable, (Tsirukis, 1989). The following example was solved by all three networks.

$$minimize \quad f(x) = -x_1 x_2^2 x_3^3 / 81$$

$$subject \quad to$$

$$h_1(x) = x_1^3 + x_2^2 + x_3 - 13 = 0$$

$$h_2(x) = x_2^2 x_3^{-1/2} - 1 = 0$$

Convergence was achieved by all the networks starting from both feasible and infeasible initial points. Figures 2 and 3 depict the algorithmic superiority of the SQP network.

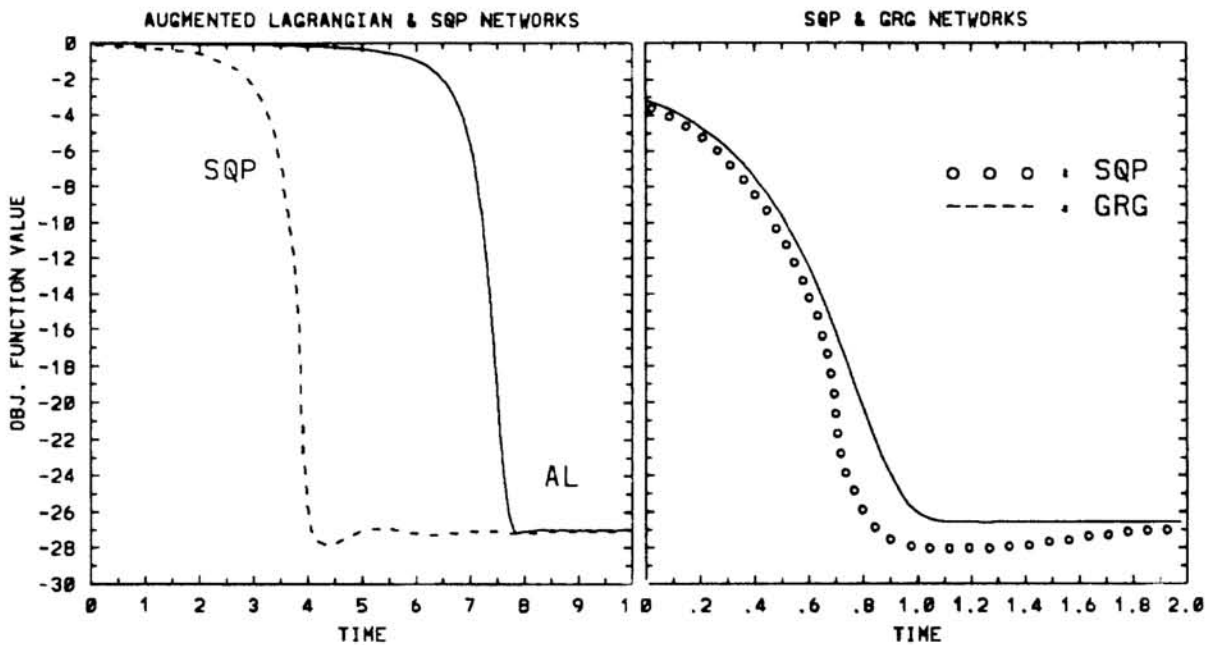

**Figure 2.** Feasible Initial State.    **Figure 3.** Infeasible Initial State.

# 6 OPTIMIZATION & PARALLEL COMPUTATION

The presented model can be directly translated into a parallel nonlinear optimizer - nonlinear equation solver - which efficiently distributes the computational burden to a large number of digital processors (at most N+K). Each one of them corresponds to an optimization variable, continuously updated by numerically integrating the state equations:

$$x_j^{(t+1)} = \phi(x^{(t)}, x^{(t+1)})$$

where $\phi$ depends on the optimization algorithm and the integration method. After each update the new value is communicated to the network.

The presented algorithm has some unique features: The state equations are differentials of the same function, the Lagrangian. Therefore, a simple integration method (eg. explicit) can be used for the steady-state computation. Also, the integration in each processor can be done asynchronously, independent of the state of the other processors. Thus, the algorithm is robust to intercommunication and execution delays.

## Acknowledgements

An extended version of this work has appeared in (Tsirukis, 1990). The authors wish to thank M.I.T. Press Journals for their permission to publish it in the present form.

## Footnotes

[1] To whom correspondence should be addressed.

## References

Bruck, J. and J. Goodman (1988). *On the Power of Neural Networks for Solving Hard Problems*. Neural Information Processing Systems, D.Z. Anderson (ed.), American Institute of Physics, New York, NY, 137-143.

Hopfield J.J. (1984), *Neurons with Graded Response have Collective Computational Properties like those of Two-state Neurons*, Proc. Natl. Acad. Sci. USA, vol. 81, 3088-3092.

Jeffrey, W. and R. Rosner (1986), *Neural Network Processing as a Tool for Function Optimization*, Neural Networks for Computing. J.S. Denker (ed.), American Institute of Physics, New York, NY, 241-246.

Kennedy, M.P. and L.O. Chua (1988), *Neural Networks for Nonlinear Programming*, IEEE Transactions on Circuits and Systems, vol. 35, no. 5, pp. 554-562.

Reklaitis, G.V., A. Ravindran and K.M. Ragsdell (1983), *Engineering Optimization: Methods and Applications*, Wiley - Interscience.

Tank, D.W. and J.J. Hopfield (1986), *Simple "Neural" Optimization Networks: An A/D Converter, Signal Decision Circuit, and a Linear Programming Circuit*, IEEE Transactions on circuits and systems, CAS-33, no. 5.

Tsirukis, A. G., Reklaitis, G.V., and Tenorio, M.F. (1989). *Computational properties of Generalized Hopfield Networks applied to Nonlinear Optimization*. Tech. Rep. TREE 89-69, School of Electrical Engineering, Purdue University.

Tsirukis, A. G., Reklaitis, G.V., and Tenorio, M.F. (1990). *Nonlinear Optimization using Generalized Hopfield Networks*, Neural Computation, vol. 1, no. 4.
